# An Analysis of Turbo Decoding with Gaussian Densities

**Paat Rusmevichientong and Benjamin Van Roy**
Stanford University
Stanford, CA 94305
{*paatrus, bvr*} *@stanford.edu*

## Abstract

We provide an analysis of the turbo decoding algorithm (TDA) in a setting involving Gaussian densities. In this context, we are able to show that the algorithm converges and that – somewhat surprisingly – though the density generated by the TDA may differ significantly from the desired posterior density, the means of these two densities coincide.

## 1 Introduction

In many applications, the state of a system must be inferred from noisy observations. Examples include digital communications, speech recognition, and control with incomplete information. Unfortunately, problems of inference are often intractable, and one must resort to approximation methods. One approximate inference method that has recently generated spectacular success in certain coding applications is the turbo decoding algorithm [1, 2], which bears a close resemblance to message–passing algorithms developed in the coding community a few decades ago [4]. It has been shown that the TDA is also related to well–understood exact inference algorithms [5, 6], but its performance on the intractable problems to which it is applied has not been explained through this connection.

Several other papers have further developed an understanding of the turbo decoding algorithm. The exact inference algorithms to which turbo decoding has been related are variants of belief propagation [7]. However, this algorithm is designed for inference problems for which graphical models describing conditional independencies form trees, whereas graphical models associated with turbo decoding possess many loops. To understand the behavior of belief propagation in the presence of loops, Weiss has analyzed the algorithm for cases where only a single loop is present [11]. Other analyses that have shed significant light on the performance of the TDA in its original coding context include [8, 9, 10].

In this paper, we develop a new line of analysis for a restrictive setting in which underlying distributions are Gaussian. In this context, inference problems are tractable and the use of approximation algorithms such as the TDA are unnecessary. However, studying the TDA in this context enables a streamlined analysis that generates new insights into its behavior. In particular, we will show that the algorithm converges and that the mean of the resulting distribution coincides with that of the

desired posterior distribution.

While preparing this paper, we became aware of two related initiatives, both involving analysis of belief propagation when priors are Gaussian and graphs possess cycles. Weiss and Freeman [12] were studying the case of graphs possessing only cliques of size two. Here, they were able to show that, if belief propagation converges, the mean of the resulting approximation coincides with that of the true posterior distribution. At the same time, Frey [3] studied a case involving graphical structures that generalize those employed in turbo decoding. He also conducted an empirical study.

The paper is organized as follows. In Section 2, we provide our working definition of the TDA. In Section 3, we analyze the case of Gaussian densities. Finally, a discussion of experimental results and open issues is presented in Section 4.

## 2   A Definition of Turbo Decoding

Consider a random variable $x$ taking on values in $\Re^n$ distributed according to a density $p_0$. Let $y_1$ and $y_2$ be two random variables that are conditionally independent given $x$. For example, $y_1$ and $y_2$ might represent outcomes of two independent transmissions of the signal $x$ over a noisy communication channel. If $y_1$ and $y_2$ are observed, then one might want to infer a posterior density $f$ for $x$ conditioned on $y_1$ and $y_2$. This can be obtained by first computing densities $p_1^*$ and $p_2^*$, where the first is conditioned on $y_1$ and the second is conditioned on $y_2$. Then,

$$f = \alpha \left( \frac{p_1^* p_2^*}{p_0} \right),$$

where $\alpha$ is a "normalizing operator" defined by

$$\alpha g \equiv \frac{g}{\int g(\bar{x}) d\bar{x}},$$

and multiplication/division are carried out pointwise.

Unfortunately, the problem of computing $f$ is generally intractable. The computational burden associated with storing and manipulating high–dimensional densities appears to be the primary obstacle. This motivates the idea of limiting attention to densities that factor. In this context, it is convenient to define an operator $\pi$ that generates a density that factors while possessing the same marginals as another density. In particular, this operator is defined by

$$(\pi g)(a) \equiv \prod_{i=1}^{n} \int_{\{\bar{x} \in \Re^n \mid \bar{x}_i = a_i\}} g(\bar{x}) d\bar{x} \wedge d\bar{x}_i$$

for all densities $g$ and all $a \in \Re^n$, where $d\bar{x} \wedge d\bar{x}_i = d\bar{x}_1 \cdots d\bar{x}_{i-1} d\bar{x}_{i+1} \cdots d\bar{x}_n$. One may then aim at computing $\pi f$ as a proxy for $f$. Unfortunately, even this problem is generally intractable. The TDA can be viewed as an iterative algorithm for approximating $\pi f$.

Let operators $F_1$ and $F_2$ be defined by

$$F_1 g = \alpha \left( \left( \pi \frac{p_1^* g}{p_0} \right) \frac{p_0}{g} \right),$$

and

$$F_2 g = \alpha \left( \left( \pi \frac{g p_2^*}{p_0} \right) \frac{p_0}{g} \right),$$

for any density $g$. The TDA is applicable in cases where computation of these two operations is tractable. The algorithm generates sequences $q_1^{(k)}$ and $q_2^{(k)}$ according to

$$q_1^{(k+1)} = F_1 q_2^{(k)} \quad \text{and} \quad q_2^{(k+1)} = F_2 q_1^{(k)}.$$

initialized with densities $q_1^{(0)}$ and $q_2^{(0)}$ that factor. The hope is that $\alpha(q_1^{(k)} q_2^{(k)}/p_0)$ converges to an approximation of $\pi f$.

## 3 The Gaussian Case

We will consider a setting in which joint density of $x$, $y_1$, and $y_2$, is Gaussian. In this context, application of the TDA is not warranted – there are tractable algorithms for computing conditional densities when priors are Gaussian. Our objective, however, is to provide a setting in which the TDA can be analyzed and new insights can be generated.

Before proceeding, let us define some notation that will facilitate our exposition. We will write $g \sim N(\mu_g, \Sigma_g)$ to denote a Gaussian density $g$ whose mean vector and covariance matrix are $\mu_g$ and $\Sigma_g$, respectively. For any matrix $A$, $\delta(A)$ will denote a diagonal matrix whose entries are given by the diagonal elements of $A$. For any diagonal matrices $X$ and $Y$, we write $X \leq Y$ if $X_{ii} \leq Y_{ii}$ for all $i$. For any pair of nonsingular covariance matrices $\Sigma_u$ and $\Sigma_v$ such that $\Sigma_u^{-1} + \Sigma_v^{-1} - I$ is nonsingular, let a matrix $A_{\Sigma_u, \Sigma_v}$ be defined by

$$A_{\Sigma_u, \Sigma_v} \equiv (\Sigma_u^{-1} + \Sigma_v^{-1} - I)^{-1}.$$

To reduce notation, we will sometimes denote this matrix by $A_{uv}$.

When the random variables $x$, $y_1$, and $y_2$ are jointly Gaussian, the densities $p_1^*$, $p_2^*$, $f$, and $p_0$ are also Gaussian. We let

$$p_1^* \sim N(\mu_1, \Sigma_1), \quad p_2^* \sim N(\mu_2, \Sigma_2), \quad f \sim N(\mu, \Sigma),$$

and assume that both $\Sigma_1$ and $\Sigma_2$ are symmetric positive definite matrices. We will also assume that $p_0 \sim N(0, I)$ where $I$ is the identity matrix. It is easy to show that $A_{\Sigma_1, \Sigma_2}$ is well–defined.

The following lemma provides formulas for the means and covariances that arise from multiplying and rescaling Gaussian densities. The result follows from simple algebra, and we state it without proof.

**Lemma 1** *Let $u \sim N(\mu_u, \Sigma_u)$ and $v \sim N(\mu_v, \Sigma_v)$, where $\Sigma_u$ and $\Sigma_v$ are positive definite. If $\Sigma_u^{-1} + \Sigma_v^{-1} - I$ is positive definite then*

$$\alpha \left( \frac{uv}{p_0} \right) \sim N \left( A_{uv} \left( \Sigma_u^{-1} \mu_u + \Sigma_v^{-1} \mu_v \right), A_{uv} \right).$$

One immediate consequence of this lemma is an expression for the mean of $f$:

$$\mu = A_{\Sigma_1, \Sigma_2} \left( \Sigma_1^{-1} \mu_1 + \Sigma_2^{-1} \mu_2 \right).$$

Let $\mathcal{S}$ denote the set of covariance matrices that are diagonal and positive definite. Let $\mathcal{G}$ denote the set of Gaussian densities with covariance matrices in $\mathcal{S}$. We then have the following result, which we state without proof.

**Lemma 2** *The set $\mathcal{G}$ is closed under $F_1$ and $F_2$.*

If the TDA is initialized with $q_1^{(0)}, q_2^{(0)} \in \mathcal{G}$, this lemma allows us to represent all iterates using appropriate mean vectors and covariance matrices.

### 3.1  Convergence Analysis

Under suitable technical conditions, it can be shown that the sequence of mean vectors and covariance matrices generates by the TDA converges. Due to space limitations, we will only present results pertinent to the convergence of covariance matrices. Furthermore, we will only present certain central components of the analyses. For more complete results and detailed analyses, we refer the reader to our upcoming full–length paper.

Recall that the TDA generates sequences $q_1^{(k)}$ and $q_2^{(k)}$ according to

$$q_1^{(k+1)} = F_1 q_2^{(k)} \quad \text{and} \quad q_2^{(k+1)} = F_2 q_1^{(k)}.$$

As discussed earlier, if the algorithm is initialized with elements of $\mathcal{G}$, by Lemma 2,

$$q_1^{(k)} \sim N\left(m_1^{(k)}, \Sigma_1^{(k)}\right) \quad \text{and} \quad q_2^{(k)} \sim N\left(m_2^{(k)}, \Sigma_2^{(k)}\right),$$

for appropriate sequences of mean vectors and covariance matrices. It turns out that there are mappings $T_1 : \mathcal{S} \mapsto \mathcal{S}$ and $T_2 : \mathcal{S} \mapsto \mathcal{S}$ such that

$$\Sigma_1^{(k+1)} = T_1\left(\Sigma_2^{(k)}\right) \quad \text{and} \quad \Sigma_2^{(k+1)} = T_2\left(\Sigma_1^{(k)}\right),$$

for all $k$. Let $T \equiv T_1 \circ T_2$. To establish convergence of $\Sigma_1^{(k)}$ and $\Sigma_2^{(k)}$, it suffices to show that $T^n(\Sigma_2^{(0)})$ converges. The following theorem establishes this and further points out that the limit does not depend on the initial iterates.

**Theorem 1** *There exists a matrix $V^* \in \mathcal{S}$ such that*

$$\lim_{n \to \infty} T^n(V) = V^*,$$

*for all $V \in \mathcal{S}$.*

#### 3.1.1  Preliminary Lemmas

Our proof of Theorem 1 relies on a few lemmas that we will present in this section. We begin with a lemma that captures important abstract properties of the function $T$. Due to space constraints, we omit the proof, even though it is nontrivial.

**Lemma 3**
(a) *There exists a matrix $\overline{D} \in \mathcal{S}$ such that for all $D \in \mathcal{S}$, $\overline{D} \leq T(D) \leq I$.*
(b) *For all $X, Y \in \mathcal{S}$, if $X \leq Y$ then $T(X) \leq T(Y)$.*
(c) *The function $T$ is continuous on $\mathcal{S}$.*
(d) *For all $\beta \in (0,1)$ and $D \in \mathcal{S}$, $(\beta + \alpha)T(D) \leq T(\beta D)$ for some $\alpha > 0$.*

The following lemma establishes convergence when the sequence of covariance matrices is initialized with the identity matrix.

**Lemma 4** *The sequence $T^n(I)$ converges in $\mathcal{S}$ to a fixed point of $T$.*

*Proof:* By Lemma 3(a), $T(I) \leq I$, and it follows from monotonicity of $T$ (Lemma 3(b)) that $T^{n+1}(I) \leq T^n(I)$ for all $n$. Since $T^n(I)$ is bounded below by a matrix $\overline{D} \in \mathcal{S}$, the sequence converges in $\mathcal{S}$. The fact that the limit is a fixed point of $T$ follows from the continuity of $T$ (Lemma 3(c)).  ∎

Let $V^* = \lim_{n \to \infty} T^n(I)$. This matrix plays the following special role.

**Lemma 5** *The matrix $V^*$ is the unique fixed point in $\mathcal{S}$ of $T$.*

*Proof:* Because $T^n(I)$ converges to $V^*$ and $T$ is monotonic, no matrix $V \in S$ with $V \neq V^*$ and $V^* \leq V \leq I$ can be a fixed point. Furthermore, by Lemma 3(a), no matrix $V \in S$ with $V \geq I$ and $V \neq I$ can be a fixed point. For any $V \in S$ with $V \leq V^*$, let

$$\beta_V = \sup \left\{ \beta \in (0,1] \middle| \beta V^* \leq V \right\}.$$

For any $V \in S$ with $V \neq V^*$ and $V \leq V^*$, we have $\beta_V < 1$. For such a $V$, by Lemma 3(d), there is an $\alpha > 0$ such that $T(\beta_V V^*) \geq (\beta_V + \alpha)V^*$, and therefore $T(V) \neq V$. The result follows. ∎

### 3.1.2 Proof of Theorem 1

*Proof:* For $V \in S$ with $V^* \leq V \leq I$ convergence to $V^*$ follows from Lemma 4 and monotonicity (Lemma 3(b)). For $V \in S$ with $V \geq I$, convergence follows from the fact that $V^* \leq T(V) \leq I$, which is a consequence of the two previously invoked lemmas together with Lemma 3(a).

Let us now address the case of $V \in S$ with $V \leq V^*$. Let $\beta_V$ be defined as in the proof of Lemma 5. Then, $\beta_V V^* \leq T(\beta_V V^*)$. By monotonicity, $T^n(\beta_V V^*) \leq T^{n+1}(\beta_V V^*) \leq V^*$ for all $n$. It follows that $T^n(\beta_V V^*)$ converges, and since $T$ is continuous, the limit must be the unique fixed point $V^*$. We have established convergence for elements $V$ of $S$ satisfying $V \leq V^*$ or $V \geq V^*$. For other elements of $S$, convergence follows from the monotonicity of $T$. ∎

### 3.2 Analysis of the Fixed Point

As discussed in the previous section, under suitable conditions, $F_1 \circ F_2$ and $F_2 \circ F_1$ each possess a unique fixed point, and the TDA converges on these fixed points. Let $q_1^* \sim N(\mu_{q_1^*}, \Sigma_{q_1^*})$ and $q_2^* \sim N(\mu_{q_2^*}, \Sigma_{q_2^*})$ denote the fixed points of $F_1 \circ F_2$ and $F_2 \circ F_1$, respectively. Based on Theorem 1, $\Sigma_{q_1^*}$ and $\Sigma_{q_2^*}$ are in $S$.

The following lemma provides an equation relating means associated with the fixed points. It is not hard to show that $A_{q_1^* q_2^*}$, $A_{\Sigma_1, \Sigma_{q_2^*}}$, and $A_{\Sigma_{q_1^*}, \Sigma_2}$, which are used in the statement, are well–defined.

### Lemma 6

$$A_{q_1^* q_2^*} \left( \Sigma_{q_1^*}^{-1} \mu_{q_1^*} + \Sigma_{q_2^*}^{-1} \mu_{q_2^*} \right) = A_{\Sigma_1, \Sigma_{q_2^*}} \left( \Sigma_1^{-1} \mu_1 + \Sigma_{q_2^*}^{-1} \mu_{q_2^*} \right) = A_{\Sigma_{q_1^*}, \Sigma_2} \left( \Sigma_{q_1^*}^{-1} \mu_{q_1^*} + \Sigma_2^{-1} \mu_2 \right)$$

*Proof:* It follows from the definitions of $F_1$ and $F_2$ that, if $q_1^* = F_1 q_2^*$ and $q_2^* = F_2 q_1^*$,

$$\alpha \frac{q_1^* q_2^*}{p_0} = \alpha \pi \frac{p_1^* q_2^*}{p_0} = \alpha \pi \frac{q_1^* p_2^*}{p_0}.$$

The result then follows from Lemma 1 and the fact that $\pi$ does not alter the mean of a distribution. ∎

We now prove a central result of this paper: the mean of the density generated by the TDA coincides with the mean $\mu$ of the desired posterior density $f$.

### Theorem 2 $\alpha(q_1^* q_2^* / p_0) \sim N(\mu, A_{q_1^* q_2^*})$

*Proof:* By Lemma 1, $\mu = A_{\Sigma_1, \Sigma_2} \left( \Sigma_1^{-1} \mu_1 + \Sigma_2^{-1} \mu_2 \right)$, while the mean of $\alpha(q_1^* q_2^* / p_0)$ is $A_{q_1^* q_2^*} \left( \Sigma_{q_1^*}^{-1} \mu_{q_1^*} + \Sigma_{q_2^*}^{-1} \mu_{q_2^*} \right)$. We will show that these two expressions are equal.

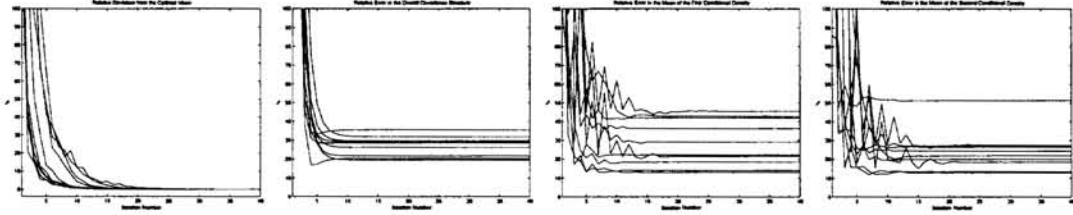

Figure 1: Evolution of errors.

Multiplying the equations from Lemma 6 by appropriate matrices, we obtain

$$A_{q_1^* q_2^*} A_{\Sigma_1, \Sigma_{q_2^*}}^{-1} A_{q_1^* q_2^*} \left( \Sigma_{q_1^*}^{-1} \mu_{q_1^*} + \Sigma_{q_2^*}^{-1} \mu_{q_2^*} \right) = A_{q_1^* q_2^*} \left( \Sigma_1^{-1} \mu_1 + \Sigma_{q_2^*}^{-1} \mu_{q_2^*} \right),$$

and

$$A_{q_1^* q_2^*} A_{\Sigma_{q_1^*}, \Sigma_2}^{-1} A_{q_1^* q_2^*} \left( \Sigma_{q_1^*}^{-1} \mu_{q_1^*} + \Sigma_{q_2^*}^{-1} \mu_{q_2^*} \right) = A_{q_1^* q_2^*} \left( \Sigma_{q_1^*}^{-1} \mu_{q_1^*} + \Sigma_2^{-1} \mu_2 \right).$$

It follows that

$$\left( A_{q_1^* q_2^*} (A_{\Sigma_1, \Sigma_{q_2^*}}^{-1} + A_{\Sigma_{q_1^*}, \Sigma_2}^{-1}) - I \right) A_{q_1^* q_2^*} \left( \Sigma_{q_1^*}^{-1} \mu_{q_1^*} + \Sigma_{q_2^*}^{-1} \mu_{q_2^*} \right) = A_{q_1^* q_2^*} \left( \Sigma_1^{-1} \mu_1 + \Sigma_2^{-1} \mu_2 \right),$$

and therefore

$$\left( A_{\Sigma_1, \Sigma_{q_2^*}}^{-1} + A_{\Sigma_{q_1^*}, \Sigma_2}^{-1} - A_{q_1^* q_2^*}^{-1} \right) A_{q_1^* q_2^*} \left( \Sigma_{q_1^*}^{-1} \mu_{q_1^*} + \Sigma_{q_2^*}^{-1} \mu_{q_2^*} \right) = \Sigma_1^{-1} \mu_1 + \Sigma_2^{-1} \mu_2.$$

Note that $A_{\Sigma_1, \Sigma_{q_2^*}}^{-1} + A_{\Sigma_{q_1^*}, \Sigma_2}^{-1} - A_{q_1^* q_2^*}^{-1} = A_{\Sigma_1, \Sigma_2}^{-1}$. It follows that

$$A_{q_1^* q_2^*} \left( \Sigma_{q_1^*}^{-1} \mu_{q_1^*} + \Sigma_{q_2^*}^{-1} \mu_{q_2^*} \right) = A_{\Sigma_1, \Sigma_2} (\Sigma_1^{-1} \mu_1 + \Sigma_2^{-1} \mu_2) = \mu.$$

∎

## 4   Discussion and Experimental Results

The limits of convergence $q_1^*$ and $q_2^*$ of the TDA provide an approximation $\alpha(q_1^* q_2^* / p_0)$ to $\pi f$. We have established that the mean of this approximation coincides with that of the desired density. One might further expect that the covariance matrix of $\alpha(q_1^* q_2^* / p_0)$ approximates that of $\pi f$, and even more so, that $q_1^*$ and $q_2^*$ bear some relation to $p_1^*$ and $p_2^*$. Unfortunately, as will be illustrated by experimental results in this section, such expectations appear to be inaccurate.

We performed experiments involving 20 and 50 dimensional Gaussian densities (i.e., $x$ was either 20 or 50 dimensional in each instance). Problem instances were sampled randomly from a fixed distribution. Due to space limitations, we will not describe the tedious details of the sampling mechanism.

Figure 1 illustrates the evolution of certain "errors" during representative runs of the TDA on 20–dimensional problems. The first graph plots relative errors in means of densities $\alpha(q_1^{(n)} q_2^{(n)} / p_0)$ generated by iterates of the TDA. As indicated by our analysis, these errors converge to zero. The second chart plots a measure of relative error for the covariance of $\alpha(q_1^{(n)} q_2^{(n)} / p_0)$ versus that of $\pi f$ for representative runs. Though these covariances converge, the ultimate errors are far from zero. The two

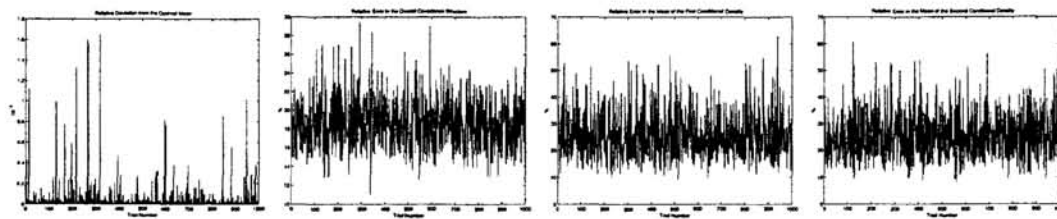

Figure 2: Errors after 50 iterations.

final graphs plot errors between the means of $q_1^{(n)}$ and $q_2^{(n)}$ and those of $p_1^*$ and $p_2^*$, respectively. Again, though these means converge, the ultimate errors can be large.

Figure 2 provides plots of the same sorts of errors measured on 1000 different instances of 50–dimensional problems after the 50th iteration of the TDA. The horizontal axes are labeled with indices of the problem instances. Note that the errors in the first graph are all close to zero (the units on the vertical axis must be multiplied by $10^{-5}$ and errors are measured in relative terms). On the other hand, errors in the other graphs vary dramatically.

It is intriguing that – at least in the context of Gaussian densities – the TDA can effectively compute conditional means without accurately approximating conditional densities. It is also interesting to note that, in the context of communications, the objective is to choose a code word $\bar{x}$ that is comes close to the transmitted code $x$. One natural way to do this involves assigning to $\bar{x}$ the code word that maximizes the conditional density $f$, i.e., the one that has the highest chance of being correct. In the Gaussian case that we have studied, this corresponds to the mean of $f$ – a quantity that is computed correctly by the TDA! It will be interesting to explore generalizations of the line of analysis presented in this paper to other classes of densities.

# References

[1] S. Benedetto and G. Montorsi, "Unveiling turbo codes: Some results on parallel concatenated coding schemes," in *IEEE Trans. Inform. Theory*, vol. 42, pp. 409-428, Mar. 1996.

[2] G. Berrou, A. Glavieux, and P. Thitimajshima, "Near Shannon limit error-correcting coding: Turbo codes," in *Proc. 1993 Int. Conf. Commun.*, Geneva, Switzerland, May 1993, pp. 1064-1070.

[3] B. Frey, "Turbo Factor Analysis." To appear in *Advances in Neural Information Processing Systems 12*.

[4] R. G. Gallager, *Low–Density Parity–Check Codes*. Cambridge, MA: MIT Press, 1963.

[5] F. R. Kschischang and B. J. Frey, "Iterative Decoding of Compound Codes by Probability Propagation in Graphical Models," in *IEEE Journal on Selected Areas in Commun.*, vol. 16, 2, pp. 219-230, Feb. 1998.

[6] R. J. McEliece, D. J. C. MacKay, and J-F. Cheng, "Turbo Decoding as an Instance of Pearl's "Belief Propagation" Algorithm," in *IEEE Journal on Selected Areas in Commun.*, vol. 16, 2, pp. 140-152, Feb. 1998.

[7] J. Pearl, *Probabilistic Reasoning in Intelligent Systems: Networks of Plausible Inference*. San Mateo, CA: Morgan Kaufmann, 1988.

[8] T. Richardson, "The Geometry of Turbo-Decoding Dynamics," Dec. 1998. To appear in *IEEE Trans. Inform. Theory*.

[9] T. Richardson and R. Urbanke, "The Capacity of Low-Density Parity Check Codes under Message-Passing Decoding", submitted to the *IEEE Trans. on Information Theory*.

[10] T. Richardson, A. Shokrollahi, and R. Urbanke, "Design of Provably Good Low-Density Parity Check Codes," submitted to the *IEEE Trans. on Information Theory*.

[11] Y. Weiss, "Belief Propagation and Revision in Networks with Loops," November 1997. Available by ftp to publications.ai.mit.edu.

[12] Y. Weiss and W. T. Freeman, "Correctness of belief propagation in Gaussian graphical models of arbitrary topology." To appear in *Advances in Neural Information Processing Systems 12*.
